# From Weighted Classification to Policy Search

**D. Blatt**
Department of Electrical Engineering
and Computer Science
University of Michigan
Ann Arbor, MI 48109-2122
dblatt@eecs.umich.edu

**A. O. Hero**
Department of Electrical Engineering
and Computer Science
University of Michigan
Ann Arbor, MI 48109-2122
hero@eecs.umich.edu

## Abstract

This paper proposes an algorithm to convert a $T$-stage stochastic decision problem with a continuous state space to a sequence of supervised learning problems. The optimization problem associated with the trajectory tree and random trajectory methods of Kearns, Mansour, and Ng, 2000, is solved using the Gauss-Seidel method. The algorithm breaks a multi-stage reinforcement learning problem into a sequence of single-stage reinforcement learning subproblems, each of which is solved via an exact reduction to a weighted-classification problem that can be solved using off-the-self methods. Thus the algorithm converts a reinforcement learning problem into simpler supervised learning subproblems. It is shown that the method converges in a finite number of steps to a solution that cannot be further improved by componentwise optimization. The implication of the proposed algorithm is that a plethora of classification methods can be applied to find policies in the reinforcement learning problem.

## 1 Introduction

There has been increased interest in applying tools from supervised learning to problems in reinforcement learning. The goal is to leverage techniques and theoretical results from supervised learning for solving the more complex problem of reinforcement learning [3]. In [6] and [4], classification was incorporated into approximate policy iterations. In [2], regression and classification are used to perform dynamic programming. Bounds on the performance of a policy which is built from a sequence of classifiers were derived in [8] and [9].

Similar to [8], we adopt the generative model assumption of [5] and tackle the problem of finding good policies within an infinite class of policies, where performance is evaluated in terms of empirical averages over a set of trajectory trees. In [8] the T-step reinforcement learning problem was converted to a set of weighted classification problems by trying to fit the classifiers to the maximal path on the trajectory tree of the decision process.

In this paper we take a different approach. We show that while the task of finding the global optimum within a class of non-stationary policies may be overwhelming, the componentwise search leads to single step reinforcement learning problems which can be reduced to a sequence of weighted classification problems. Our reduction is exact and is differ-

ent from the one proposed in [8]; it gives more weight to regions of the state space in which the difference between the possible actions in terms of future reward is large, rather than giving more weight to regions in which the maximal future reward is large. The weighted classification problems can be solved by applying weights-sensitive classifiers or by further reducing the weighted classification problem to a standard classification problem using re-sampling methods (see [7], [1], and references therein for a description of both approaches). Based on this observation, an algorithm that converts the policy search problem into a sequence of weighted classification problems is given. It is shown that the algorithm converges in a finite number of steps to a solution, which cannot be further improved by changing the control of a single stage while holding the rest of the policy fixed.

## 2    Problem Formulation

The results are presented in the context of MDPs but can be applied to POMDPs and non-Markovian decision processes as well. Consider a T-step MDP $\mathcal{M} = \{\mathcal{S}, \mathcal{A}, D, P_{s,a}\}$, where $\mathcal{S}$ is a (possibly continuous) state space, $\mathcal{A} = \{0, \ldots, L-1\}$ is a finite set of possible actions, $D$ is the distribution of the initial state, and $P_{s,a}$ is the distribution of the next state given that the current state is $s$ and the action taken is $a$. The reward granted when taking action $a$ at state $s$ and making a transition to state $s'$ is assumed to be a known deterministic and bounded function of $s'$ denoted by $r : \mathcal{S} \to [-M, M]$. No generality is lost in specifying a known deterministic reward since it is possible to augment the state variable by an additional random component whose distribution depends on the previous state and action, and specify the function $r$ to extract this random component. Denote by $S_0, S_1, \ldots, S_T$ the random state variables.

A non-stationary deterministic policy $\pi = (\pi_0, \pi_1, \ldots, \pi_{T-1})$ is a sequence of mappings $\pi_t : \mathcal{S} \to \mathcal{A}$, which are called controls. The control $\pi_t$ specifies the action taken at time $t$ as a function of the state at time $t$. The expected sum of rewards of a non-stationary deterministic policy $\pi$ is given by

$$V(\pi) = \mathrm{E}_\pi \left\{ \sum_{t=1}^{T} r(S_t) \right\}, \tag{1}$$

where the expectation is taken with respect to the distribution over the random state variables induced by the policy $\pi$. We call $V(\pi)$ the value of policy $\pi$. Non-stationary deterministic policies are considered since the optimal policy for a finite horizon MDP is non-stationary and deterministic [10]. Usually the optimal policy is defined as the policy that maximizes the value conditioned on the initial state, i.e.,

$$V_\pi(s) = \mathrm{E}_\pi \left\{ \sum_{t=1}^{T} R(S_t) | S_0 = s \right\}, \tag{2}$$

for any realization $s$ of $S_0$ [10]. The policy that maximizes the conditional value given each realization of the initial state also maximizes the value averaged over the initial state, and it is the unique maximizer if the distribution of the initial state $D$ is positive over $\mathcal{S}$. Therefore, when optimizing over all possible policies, the maximization of (1) and (2) are equivalent. When optimizing (1) over a restricted class of policies, which does not contain the optimal policy, the distribution over the initial state specifies the importance of different regions of the state space in terms of the approximation error. For example, assigning high probability to a certain region of $\mathcal{S}$ will favor policies that well approximate the optimal policy over that region. Alternatively, maximizing (1) when $D$ is a point mass at state $s$ is equivalent to maximizing (2).

Following the generative model assumption of [5], the initial distribution $D$ and the conditional distribution $P_{s,a}$ are unknown but it is possible to generate realization of the initial

state according to $D$ and the next state according to $P_{s,a}$ for arbitrary state-action pairs $(s, a)$. Given the generative model, $n$ trajectory trees are constructed in the following manner. The root of each tree is a realization of $S_0$ generated according to the distribution $D$. Given the realization of the initial state, realizations of the next state $S_1$ given the $L$ possible actions, denoted by $S_1|a$, $a \in \mathcal{A}$, are generated. Note that this notation omits the dependence on the value of the initial state. Each of the $L$ realizations of $S_1$ is now the root of the subtree. These iterations continue to generate a depth $T$ tree. Denote by $S_t|i_0, i_1, \ldots, i_{t-1}$ the random variable generated at the node that follows the sequence of act̶ ̶ ̶ ̶ ̶ ̶ ̶ ̶ ̶ ̶ ̶ ̶ ̶ ̶ ̶ ̶ ̶ ̶ ̶ ̶ ̶ ̶ ̶ ̶ ̶ ̶ ̶ ̶ ̶ ̶ ̶ ̶ e
ge̶

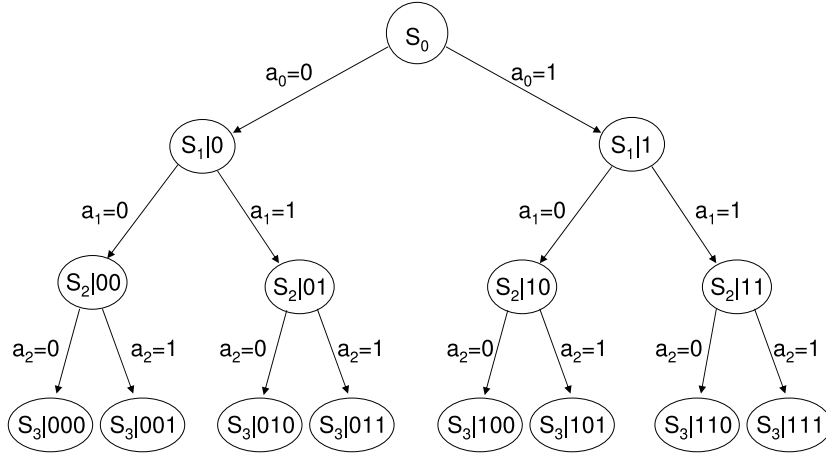

Figure 1: A binary trajectory tree.

Consider a class of policies $\Pi$, i.e., each element of $\Pi$ is a sequence of $T$ mappings from $\mathcal{S}$ to $\mathcal{A}$. It is possible to estimate the value of any policy in the class from the set of trajectory trees by simply averaging the sum of rewards on each tree along the path that agrees with the policy [5]. Denote by $\widehat{V}^i(\pi)$ the observed value on the $i$'th tree along the path that corresponds to the policy $\pi$. Then the value of the policy $\pi$ is estimated by

$$\widehat{V}_n(\pi) = n^{-1} \sum_{i=1}^{n} \widehat{V}^i(\pi). \tag{3}$$

In [5], the authors show that with high probability (over the data set) $\widehat{V}_n(\pi)$ converges uniformly to $V(\pi)$ (1) with rates that depend on the VC-dimension of the policy class. This result motivates the use of policies $\pi$ with high $\widehat{V}_n(\pi)$, since with high probability these policies have high values of $V(\pi)$. In this paper, we consider the problem of finding policies that obtain high values of $\widehat{V}_n(\pi)$.

## 3 A Reduction From a Single Step Reinforcement Learning Problem to Weighted Classification

The building block of the proposed algorithm is an exact reduction from a single step reinforcement learning to a weighted classification problem. Consider the single step decision process. An initial state $S_0$ generated according to the distribution $D$ is followed by one of L possible actions $A \in \{0, 1, \ldots, L-1\}$, which leads to a transition to state $S_1$ whose

conditional distribution given the initial state is $s$ and the action is $a$ is given by $P_{s,a}$. Given a class of policies $\Pi$, where policy in $\Pi$ is a map from $\mathcal{S}$ to $\mathcal{A}$, the goal is to find

$$\widehat{\pi} \in \arg\max_{\pi \in \Pi} \widehat{V}_n(\pi). \tag{4}$$

In this single step problem the data are $n$ realization of the random element $\{S_0, S_1|0, S_1|1, \ldots, S_1|L-1\}$. Denote the $i$'th realization by $\{s_0^i, s_1^i|0, s_1^i|1, \ldots, s_1^i|L-1\}$. In this case, $\widehat{V}_n(\pi)$ can be written explicitly by

$$\widehat{V}_n(\pi) = \mathrm{E}_n \left\{ \sum_{l=0}^{L-1} r(S_1|l) I(\pi(S_0) = l) \right\}, \tag{5}$$

where for a function $f$, $\mathrm{E}_n \{f(S_0, S_1|0, S_1|1, \ldots, S_1|L-1)\}$ is its empirical expectation $n^{-1} \sum_{i=1}^n f(s_0^i, s_1^i|0, s_1^i|1, \ldots, s_1^i|L-1)$, and $I(\cdot)$ is the indicator function taking a value of one when its argument is true and zero otherwise.

The following proposition shows that the problem of maximizing the empirical reward (5) is equivalent to a weighted classification problem.

**Proposition 1** *Given a class of policies $\Pi$ and a set of $n$ trajectory trees,*

$$\arg\max_{\pi \in \Pi} \mathrm{E}_n \left\{ \sum_{l=0}^{L-1} r(S_1|l) I(\pi(S_0) = l) \right\}$$

$$= \arg\min_{\pi \in \Pi} \mathrm{E}_n \left\{ \sum_{l=0}^{L-1} \left[ \max_k r(S_1|k) - r(S_1|l) \right] I(\pi(S_0) = l) \right\}. \tag{6}$$

The proposition implies that the maximizer of the empirical reward over a class of policies is the output of an optimal weights dependent classifier for the data set:

$$\left\{ \left( s_0^i, \arg\max_k r(s_1^i|k), w^i \right) \right\}_{i=1}^n,$$

where for each sample, the first argument is the example, the second is the label, and

$$w^i = \left[ \max_k r(s_1^i|k) - r(s_1^i|0), \max_k r(s_1^i|k) - r(s_1^i|1), \ldots, \max_k r(s_1^i|k) - r(s_1^i|L-1) \right]$$

is the realization of the $L$ costs of classifying example $i$ to each of the possible labels. Note that the realizations of the costs are always non-negative and the cost of the correct classification ($\arg\max_k r(s_1^i|k)$) is always zero. The solution to the weighted classification problem is a map from $\mathcal{S}$ to $\mathcal{A}$ which minimizes the empirical weighted misclassification error (6). The proposition asserts that this mapping is also the control which maximizes the empirical reward (5).

**Proof 1** *For all $j \in \{0, 1, \ldots, L-1\}$,*

$$\sum_{l=0}^{L-1} r(S_1|l) I(\pi(S_0) = l) = r(S_1|j) + (r(S_1|0) - r(S_1|j)) I(\pi(s) = 0) + \tag{7}$$

$$(r(S_1|1) - r(S_1|j)) I(\pi(s) = 1) + \ldots + (r(S_1|L-1) - r(S_1|j)) I(\pi(s) = L-1).$$

*In addition,*

$$\mathrm{E}_n \left\{ \sum_{l=0}^{L-1} r(S_1|l) I(\pi(S_0) = l) \right\} =$$

$$\mathrm{E}_n\left\{I(\arg\max_k r(S_1|k)=0)\sum_{l=0}^{L-1} r(S_1|l)I(\pi(S_0)=l)\right\}+$$

$$\mathrm{E}_n\left\{I(\arg\max_k r(S_1|k)=1)\sum_{l=0}^{L-1} r(S_1|l)I(\pi(S_0)=l)\right\}+\ldots+$$

$$\mathrm{E}_n\left\{I(\arg\max_k r(S_1|k)=L-1)\sum_{l=0}^{L-1} r(S_1|l)I(\pi(S_0)=l)\right\}.$$

*Substituting (7) we obtain*

$$\mathrm{E}_n\left\{\sum_{l=0}^{L-1} r(S_1|l)I(\pi(S_0)=l)\right\}=$$

$$\sum_{j=0}^{L-1}\mathrm{E}_n\{I(\arg\max_k r(S_1|k)=j)[r(S_1|j)-$$

$$(\max_k r(S_1|k)-r(S_1|0))I(\pi(S_0)=0)-$$

$$(\max_k r(S_1|k)-r(S_1|1))I(\pi(S_0)=1)-\ldots-$$

$$(\max_k r(S_1|k)-r(S_1|L-1))I(\pi(S_0)=L-1)]\}=$$

$$\sum_{j=0}^{L-1}\mathrm{E}_n\left\{I(\arg\max_k r(S_1|k)=j)r(S_1|j)\right\}-$$

$$\mathrm{E}_n\left\{\sum_{l=0}^{L-1}\left[\max_k R(S_1|k)-R(S_1|l)\right]I(\pi(S_0)=l)\right\}$$

*The term in the second to last line is independent of $\pi(s)$ and the result follows.*

In the binary case, the optimization problem is

$$\arg\min_{\pi\in\Pi}\mathrm{E}_n\left\{|r(S_1|0)-r(S_1|1)|I(\pi(S_0)\neq\arg\max_k r(S_1|k))\right\},$$

i.e., the single step reinforcement learning problem reduces to the weighted classification problem with samples

$$\left\{\left(s_0^i,\arg\max_{k\in\{0,1\}} r(s_1^i|k),|r(s_1^i|0)-r(s_1^i|1)|\right)\right\}_{i=1}^n,$$

where for each sample, the first argument is the example, the second is the label, and the third is a realization of the cost incurred when misclassifying the example. Note that this is different from the reduction in [8]. When applying the reduction in [8] to our single step problem the costs are taken to be $\max_{k\in\{0,1\}} r(s_1^i|k)$ rather than $|r(s_1^i|0)-r(s_1^i|1)|$. Setting the costs to $\max_{k\in\{0,1\}} r(s_1^i|k)$ instead of $|r(s_1^i|0)-r(s_1^i|1)|$ favors classifiers which perform well in regions where the maximal reward is large (regardless of the difference between the two actions) instead of regions where the difference between the rewards that result from the two actions is large. It is easy to set an example of a simple MDP and a restricted class of policies, which do not include the optimal policy, in which the classifier that minimizes the weighted misclassification problem with costs $\max_{k\in\{0,1\}} r(s_1^i|k)$ is not equivalent to the optimal policy. When using our reduction, they are always equivalent. On the other hand, in [8] the choice $\max_{k\in\{0,1\}} r(s_1^i|k)$ led to a bound on the performance of the policy in terms of the performance of the classifier. We do not pursue this

type of bounds here since given the classifier, the performance of the resulting policy can be directly estimated from (5). Given a sequence of classifiers, the value of the induced sequence of controls (or policy) can be estimated directly by (3) with generalization guarantees provided by the bounds in [5]. In [2], a certain single step binary reinforcement learning problem is converted to weighted classification by averaging multiple realizations of the rewards under the two possible actions for each state. As seen here, this Monte Carlo approach is not necessary; it is sufficient to sample the rewards once for each state.

## 4 Finding Good Policies for a $T$-Step Markov Decision Processes By Solving a Sequence of Weighted Classification Problems

Given the class of policies $\Pi$, the algorithm updates the controls $\pi_0, \ldots, \pi_{T-1}$ one at a time in a cyclic manner while holding the rest constant. Each update is formulated as a single step reinforcement learning problem which is then converted to a weighted classification problem. In practice, if the weighted classification problem is only approximately solved, then the new control is accepted only if it leads to higher value of $\widehat{V}$. When updating $\pi_t$, the trees are pruned from the root to stage $t$ by keeping only the branch which agrees with the controls $\pi_0, \pi_1, \ldots, \pi_{t-1}$. Then a single step reinforcement learning is formulated at time step $t$, where the realization of the reward which follows action $a \in \mathcal{A}$ at stage $t$ is the immediate reward obtained at the state which follows action $a$ plus the sum of rewards which are accumulated along the branch which agrees with the controls $\pi_{t+1}, \pi_{t+2}, \ldots, \pi_{T-1}$. The iterations end after the first complete cycle with no parameter modifications.

Note that when updating $\pi_t$, each tree contributes one realization of the state at time $t$. A result of the pruning is that the ensemble of state realization are drawn from the distribution induced by the policy up to time $t-1$. In other words, the algorithm relaxes the requirement in [2] to have access to a baseline distribution - a distribution over the stat̶̶̶̶̶̶̶̶̶̶̶̶̶̶̶ erates samples fro̶̶̶̶̶̶̶̶̶̶̶̶̶̶̶ ng policies.

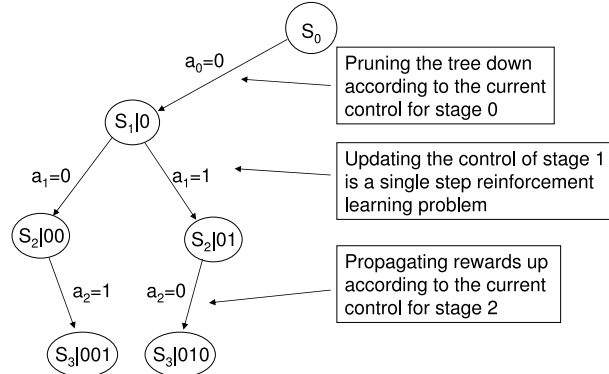

Figure 2: Updating $\pi_1$. In the example: pruning down according to $\pi_0(S_0) = 0$, propagating rewards up according to $\pi_2(S_2|00) = 1$, and $\pi_2(S_2|01) = 0$.

**Proposition 2** *The algorithm converges after a finite number of iterations to a policy that cannot be further improved by changing one of the controls and holding the rest fixed.*

**Proof 2** *Writing the empirical average sum of rewards $\widehat{V}_n(\pi)$ explicitly as*

$$\widehat{V}_n(\pi) = \mathrm{E}_n \left\{ \sum_{i_0, \ldots, i_{T-1} \in \mathcal{A}^T} I(\pi_0(S^0) = i_0) I(\pi_1(S^1|i^0) = i^1) \ldots \right.$$

$$I(\pi_{T-1}(S^{T-1}|i^0, i^1, \ldots, i^{T-2}) = i^{T-1}) \sum_{t=1}^{T} r(S^t|i^0, i^1, \ldots, i^{t-1}) \Bigg\},$$

*it can be seen that the algorithm is a Gauss-Seidel algorithm for maximizing $\widehat{V}_n(\pi)$, where, at each iteration, optimization of $\pi_t$ is carried out at one of the stages $t$ while keeping $\pi_{t'}$, $t' \neq t$ fixed. At each iteration the previous control is a valid solution and hence the objective function is non decreasing. Since $\widehat{V}_n(\pi)$ is evaluated using a finite number of trees, it can take only a finite set of values. Therefore, we must reach a cycle with no updates after a finite number of iterations. A cycle with no improvements implies that we cannot increase the empirical average sum of rewards by updating one of the $\pi_t$'s.*

## 5  Initialization

There are two possible initial policies that can be extracted from the set of trajectory trees. One possible initial policy is the myopic policy which is computed from the root of the tree downwards. Staring from the root, $\pi_0$ is found by solving the single stage reinforcement learning resulting from taking into account only the immediate reward at the next state. Once the weighted classification problem is solved the trees are pruned by following the action which agrees with $\pi_0$. The remaining realizations of state $S_1$ follow the distribution induced by the myopic control of the first stage. The process is continued to stage $T-1$. The second possible initial policy is computed from the leaves backward to the root. Note that the distribution of the state at a leaf that is chosen at random is the distribution of the state when a randomized policy is used. Therefore, to find the best control at stage $T-1$, given that the previous $T-2$ controls choose random actions, we solve the weighted classification problem induced by considering all the realization of the state $S_{T-1}$ from all the trees (these are not independent observations) or choose randomly one realization from each tree (these are independent realizations). Given the classifier, we use the equivalent control $\pi_{T-1}$ to propagated the rewards up to the previous stage and solve the resulting weighted classification problem. This is carried out recursively up to the root of the tree.

## 6  Extensions

The results presented in this paper generalize to the non-Markovian setting as well. In particular, when the state space, action space, and the reward function depend on time, and the distribution over the next state depends on all past states and actions, we will be dealing with non-stationary deterministic policies $\pi = (\pi_0, \pi_1, \ldots, \pi_{T-1})$; $\pi_t : \mathcal{S}_0 \times \mathcal{A}_0 \times \ldots \times \mathcal{S}_{t-1} \times \mathcal{A}_{t-1} \times \mathcal{S}_t \to \mathcal{A}_t$, $t = 0, 1, \ldots, T-1$. POMDPs can be dealt with in terms of the belief states as a continuous state space MDP or as a non-Markovian process in which policies depend directly on all past observations.

While we focused on the trajectory tree method, the algorithm can be easily modified to solve the optimization problem associated with the random trajectory method [5] by adjusting the single step reinforcement learning reduction and the pruning method presented here.

## 7  Illustrative Example

The following example illustrates the aspects of the problem and the components of our solution. The simulated system is a two-step MDP, with continuous state space $\mathcal{S} = [0, 1]$ and a binary action space $\mathcal{A} = \{0, 1\}$. The distribution over the initial state is uniform. Given state $s$ and action $a$ the next state $s'$ is generated by $s' = \mathrm{mod}(s + 0.33a + 0.1\mathrm{randn}, 1)$,

where $\mathrm{mod}(x,1)$ is the fraction part of $x$, and $\mathtt{randn}$ is a Gaussian random variable independent of the other variables in the problem. The reward function is $r(s) = s\sin(\pi s)$. We consider a class of policies parameterized by a continuous parameter: $\Pi = \{\pi(\cdot;\theta)|\theta = (\theta_0,\theta_1) \in [0,2]^2\}$, where $\pi_i(s;\theta_i) = 1$ when $\theta_i \leq 1$ and $s > \theta_i$ or when $\theta_i > 1$ and $s < \theta_i - 1$ and zero otherwise, $i = 0, 1$.

In Figure 3 the objective function $\widehat{V}_n(\pi(\theta))$, estimated from $n = 20$ trees, is presented as a function of $\theta_0$ and $\theta_1$. The path taken by the algorithm supperimposed on the contour plot of $\widehat{V}_n(\pi(\theta))$ is also presented. Starting from the arbitrary point 0, the algorithm performs optimization with respect to one of the coordinates at a time and converges after 3 iterations.

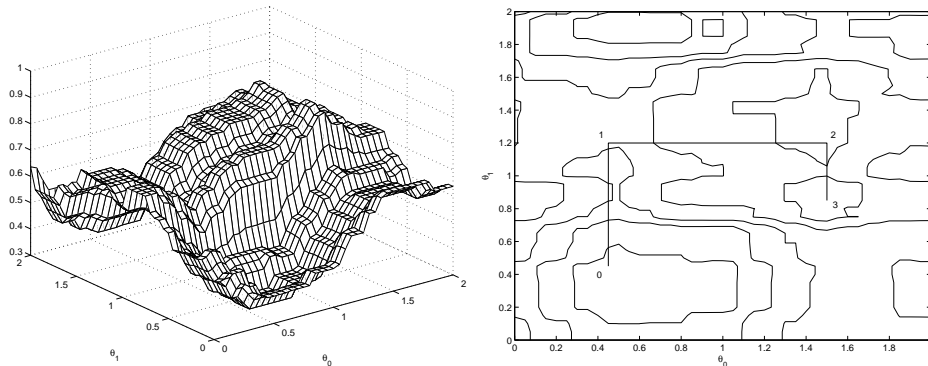

Figure 3: The objective function $\widehat{V}_n(\pi(\theta))$ and the path taken by the algorithm.

## References

[1] N. Abe, B. Zadrozny, and J. Langford. An iterative method for multi-class cost-sensitive learning. In *Proceedings of the Tenth ACM SIGKDD International Conference on Knowledge Discovery and Data Mining*, pages 3–11, 2004.

[2] J. Bagnell, S. Kakade, A. Ng, and J. Schneider. Policy search by dynamic programming. In *Advances in Neural Information Processing Systems*, volume 16. MIT Press, 2003.

[3] A. G. Barto and T. G. Dietterich. Reinforcement learning and its relationship to supervised learning. In J. Si, A. Barto, W. Powell, and D. Wunsch, editors, *Handbook of learning and approximate dynamic programming*. John Wiley and Sons, Inc, 2004.

[4] A. Fern, S. Yoon, and R. Givan. Approximate policy iteration with a policy language bias. In *Advances in Neural Information Processing Systems*, volume 16, 2003.

[5] M. Kearns, Y. Mansour, and A. Ng. Approximate planning in large POMDPs via reusable trajectories. In *Advances in Neural Information Processing Systems*, volume 12. MIT Press, 2000.

[6] M. Lagoudakis and R. Parr. Reinforcement learning as classification: Leveraging modern classifiers. In *Proceedings of the Twentieth International Conference on Machine Learning*, 2003.

[7] J. Langford and A. Beygelzimer. Sensitive error correcting output codes. In *Proceedings of the 18th Annual Conference on Learning Theory*, pages 158–172, 2005.

[8] J. Langford and B. Zadrozny. Reducing T-step reinforcement learning to classification. http://hunch.net/~jl/projects/reductions/reductions.html, 2003.

[9] J. Langford and B. Zadrozny. Relating reinforcement learning performance to classification performance. In *Proceedings of the Twenty Second International Conference on Machine Learning*, pages 473–480, 2005.

[10] M. L. Puterman. *Markov decision processes: discrete stochastic dynamic programming*. John Wiley & Sons, Inc, 1994.
